# Multi-effect Decompositions
# for Financial Data Modeling

**Lizhong Wu & John Moody**
Oregon Graduate Institute, Computer Science Dept.,
PO Box 91000, Portland, OR 97291
also at:
Nonlinear Prediction Systems,
PO Box 681, University Station, Portland, OR 97207

## Abstract

High frequency foreign exchange data can be decomposed into three components: the inventory effect component, the surprise information (news) component and the regular information component. The presence of the inventory effect and news can make analysis of trends due to the diffusion of information (regular information component) difficult.

We propose a neural-net-based, independent component analysis to separate high frequency foreign exchange data into these three components. Our empirical results show that our proposed multi-effect decomposition can reveal the intrinsic price behavior.

## 1  Introduction

Tick-by-tick, high frequency foreign exchange rates are extremely noisy and volatile, but they are not simply pure random walks (Moody & Wu 1996). The price movements are characterized by a number of "stylized facts" [1], including the following two properties: (1) short term, weak oscillations on a time scale of several ticks and (2) erratic occurrence of turbulence lasting from minutes to tens of minutes. Property (1) is most likely caused by the market makers' inventory effect (O'Hara 1995), and property (2) is due to surprise information, such as news, rumors, or major economic announcements. The price changes due to property (1) are referred to as the inventory effect component, and the changes due to property (2) are referred to as the surprise information component. The price changes due to other information is referred to as the regular information component.

Due to the inventory effect, price changes show strong negative correlations on short time scales (Moody & Wu 1995). Because of the surprise information effect, distributions of price changes are non-normal (Mandelbrot 1963). Since both the inventory effect and the surprise information effect are short term and temporary, their corresponding price components are independent of the fundamental price changes. However, their existence will seriously affect data analysis and modeling (Moody & Wu 1995). Furthermore, the most reliable component of price changes, for forecasting purposes, is the long term trend. The presence of high frequency oscillations and short periods of turbulence make it difficult to identify and predict the changes in such trends, if they occur.

In this paper, we propose a novel approach with the following price model:

$$q(t) = c_1 p_1(t) + c_2 p_2(t) + c_3 p_3(t) + \varepsilon(t) .  \tag{1}$$

In this model, $q(t)$ is the observed price series and $p_1(t)$, $p_2(t)$ and $p_3(t)$ correspond respectively to the regular information component, the surprise information component and the inventory effect component. $p_1(t)$, $p_2(t)$ and $p_3(t)$ are mutually independent and may individually be either iid or correlated. $\varepsilon(t)$ is process noise, and $c_1$, $c_2$ and $c_3$ are scale constants. Our goal is to find $p_1(t)$, $p_2(t)$ and $p_3(t)$ given $q(t)$.

The outline of the paper is as follows. We describe our approach for multi-effect decomposition in Section 2. In Section 3, we analyze the decomposed price components obtained for the high frequency foreign exchange rates and characterize their stochastic properties. We conclude and discuss the potential applications of our multi-effect decomposition in Section 4.

## 2  Multi-effect Decomposition

### 2.1  Independent Source Separation

The task of decomposing the observed price quotes into a regular information component, a surprise information component and an inventory effect component can be exactly fitted into the framework of independent source separation. Independent source separation can be described as follows:

> Assume that $X = \{x_i, i = 1, 2, \ldots, n\}$ are the sensor outputs which are some superposition of unknown independent sources $S = \{s_i, i = 1, 2, \ldots, m\}$. The task of independent source separation is to find a mapping $Y = f(X)$, so that $Y \approx AS$, where $A$ is an $m \times m$ matrix in which each row and column contains only one non-zero element.

Approaches to separate statistically-independent components in the inputs include

- Blind source separation (Jutten & Herault 1991),
- Information maximization (Linsker 1989), (Bell & Sejnowski 1995),
- Independent component analysis, (Comon 1994), (Amari, Cichocki & Yang 1996),
- Factorial coding (Barlow 1961).

All of these approaches can be implemented by artificial neural networks. The network architectures can be linear or nonlinear, multi-layer perceptrons, recurrent networks or other context sensitive networks (Pearlmutter & Parra 1997). We can choose a training criterion to minimize the energy in the output units, to maximize the information transferred in the network, to reduce the redundancies between the outputs, or to use the Edgeworth expansion or Gram-Charlier expansion of a probability distribution, which leads to an analytic expression of the entropy in terms of measurable higher order cumulants.

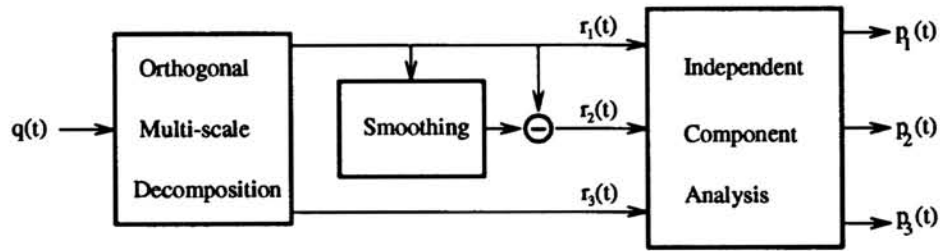

Figure 1: System diagram of multi-effect decomposition for high frequency foreign exchange rates. $q(t)$ are original price quotes, $r_i(t)$ are the reference inputs, and $p_i(t)$ are the decomposed components.

For our price decomposition problem, the non-Gaussian nature of price series requires that the transfer function of the decomposition system be nonlinear. In general, the nonlinearities in the transfer function are able to pick up higher order moments of the input distributions and perform higher order statistical redundancy reduction between outputs.

## 2.2 Reference input selection

In traditional approaches to blind source separation, nothing is assumed to be known about the inputs, and the systems adapt on-line and without a supervisor. This works only if the number of sensors is not less than the number of independent sources. If the number of sensors is less than that of sources, the sources can, in theory, be separated into disjoint groups (Cao & Liu 1996). However, the problem is ill-conditioned for most of the above practical approaches which only consider the case where the number of sensors is equal to the number of sources.

In our task to decompose the multiple components of price quotes, the problem can be divided into two cases. If the prices are sampled at regular intervals, we can use price quotes observed in different markets, and have the number of sensors be equal to the number of price components. However, in the high frequency markets, the price quotes are not regularly spaced in time. Price quotes from different markets will not appear at the same time, so we cannot apply the price quotes from different markets to the system. In this case, other reference inputs are needed.

Motivated by the use of reference inputs for noise canceling (Widrow, Glover, McCool, Kaunitz, Williams, Hearn, Zeidler, Dong & Goodlin 1975), we generate three reference inputs from original price quotes. They are the estimates of the three desired components. In the following, we briefly describe our procedure for generating the reference inputs.

By modeling the price quotes using a *"True Price"* state space model (Moody & Wu 1996)

$$q(t) = r_1(t) + r_3(t) , \qquad (2)$$

where $r_1(t)$ is an estimate of the information component (*True Price*) and $r_3(t)$ is an estimate of the inventory effect component (additive noise), and by assuming that the *True Price* $r_1(t)$ is a fractional Brownian motion (Mandelbrot & Van Ness 1968), we can estimate $r_1(t)$ and $r_3(t)$ with given $q(t)$, (Moody & Wu 1996), as

$$r_1(t) = \sum_{m,n} S(m,\theta) Q_n^m \psi_n^m(t) \qquad (3)$$

$$r_3(t) = q(t) - r_1(t) \qquad (4)$$

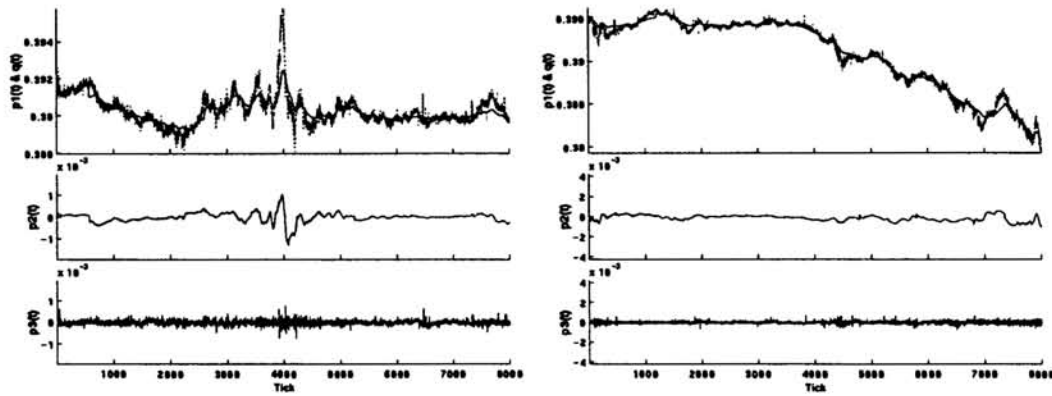

Figure 2: Multi-effect decompositions for two segments of the DEM/USD (log prices) extracted from September 1995. The three panels in each segment display the observed prices (the dotted curve in upper panel), the regular information component (solid curve in upper panel), the surprise information component (mid panel) and the inventory effect component (lower panel).

where $\psi_n^m(t)$ is an orthogonal wavelet function, $Q_n^m$ is the coefficient of the wavelet transform of $q(t)$, $m$ is the index of the scales and $n$ is the time index of the components in the wavelet transfer, $S(m, \theta)$ is a smoothing function, and its parameters can be estimated using the EM algorithm (Wornell & Oppenheim 1992).

We then estimate the surprise information component as the residual between the information component and its moving average:

$$r_2(t) = r_1(t) - s(t) \ . \tag{5}$$

$s(t)$ is an exponential moving average of $r_1(t)$ and

$$s(t) = (1 + \alpha)r_1(t) - \alpha s(t-1) \tag{6}$$

where $\alpha$ is a factor. Although it can be optimized based on the training data, we set $\alpha = -0.9$ in our current work.

Our system diagram for multi-effect decomposition is shown in Figure 1. Using multi-scale decomposition Eqn(3) and smoothing techniques Eqn(6), we obtain three reference inputs. We can then separate the reference inputs into three independent components via independent component analysis using an artificial neural network. Figure 2 presents multi-effect decompositions for two segments of the DEM/USD rates. The first segment contains some impulses, and the corresponding surprise information component is able to catch such volatile movements. The second segment is basically down trending, so its surprise information component is comparatively flat.

## 3   Empirical Analysis

### 3.1   Mutually Independent Analysis

Mutual independence of the variables is satisfied if the joint probability density function equals the product of the marginal densities, or equivalently, the characteristic function splits into the sum of marginal characteristic functions: $g(X) = \sum_{i=1}^{n} g_i(x_i)$. Taking the Taylor expansion of both sides of the above equation, products between different variables $x_i$ in the left-hand side must be zero since there are no such terms in the right-hand side.

Table 1: Comparisons between the correlation coefficients $\rho$ (normalized) and the cross-cumulants $\Gamma$ (unnormalized) of order 4 before and after independent component analysis (ICA). The DEM/USD quotes for September 1995 is divided into 147 sub-sets of 1024 ticks. The results presented here are the median values. The last column is the absolute ratio of before ICA and after ICA. We note that all ratios are greater than 1, indicating that after ICA, the components become more independent.

| Components pairs | Cross-Cumulants | Before ICA | After ICA | Absolute ratio |
|---|---|---|---|---|
| $p_1(t) \sim p_2(t)$ | $\rho_{12}$ | 0.56 | 0.14 | 4.1 |
| | $\Gamma_{13}$ | 2.7e-14 | 7.8e-17 | 342.2 |
| | $\Gamma_{22}$ | -5.6e-15 | 9.2e-16 | 6.0 |
| | $\Gamma_{31}$ | 2.0e-11 | 1.3e-13 | 148.5 |
| $p_1(t) \sim p_3(t)$ | $\rho_{13}$ | 0.15 | 0.03 | 4.7 |
| | $\Gamma_{13}$ | 2.1e-15 | 1.6e-17 | 128.9 |
| | $\Gamma_{22}$ | -2.0e-15 | -4.5e-16 | 4.5 |
| | $\Gamma_{31}$ | 5.9e-12 | 6.9e-14 | 84.5 |
| $p_2(t) \sim p_3(t)$ | $\rho_{23}$ | 0.17 | 0.04 | 4.3 |
| | $\Gamma_{13}$ | 9.1e-16 | -5.0e-19 | 1806.0 |
| | $\Gamma_{22}$ | 1.2e-15 | 4.9e-17 | 24.3 |
| | $\Gamma_{31}$ | 3.6e-15 | 3.0e-17 | 119.6 |

We observe the cross-cumulants of order 4:

$$\Gamma_{13} = M_{13} - 3M_{20}M_{11} \tag{7}$$
$$\Gamma_{22} = M_{22} - M_{20}M_{02} - 2M_{11}^2 \tag{8}$$
$$\Gamma_{31} = M_{31} - 3M_{02}M_{11} \tag{9}$$

where $M_{kl} = E\{x_i^k x_j^l\}$ denote the moments of order $k + l$. If $x_i$ and $x_j$ are independent, then their cross-cumulants must be zero (Comon 1994). Table 1 compares the cross-cumulants before and after independent component analysis (ICA) for the DEM/USD in September 1995. For reference, the correlation coefficients before and after ICA are also listed in the table. We see that after ICA, the components have become less correlated and thus more independent.

## 3.2  Autocorrelation Analysis

Figure 3 depicts the autocorrelation functions of the changes in individual components and compares them to the original returns. We compute the short-run autocorrelations for the lags up to 50. Figure 3 gives the means and standard deviations for September 1995. From the figure, we can see that both the inventory effect component and the original returns show very similar autocorrelation functions, which are dominated by the significant negative, first-order autocorrelations. The mean values for the other orders are basically equal to zero. The autocorrelations of the regular information component and the surprise information component show positive correlations except at first order. These non-zero autocorrelations are hidden by noise in the original series. The autocorrelation function of the surprise information component decays faster than that of the regular information component. On average, it is below the 95% confidence band for lags larger than 20 ticks.

The above autocorrelation analysis suggests the following. (1) Price changes due to the information effects are slightly trending on tick-by-tick time scales. The trend in the surprise information component is shorter term than that in the regular information component.

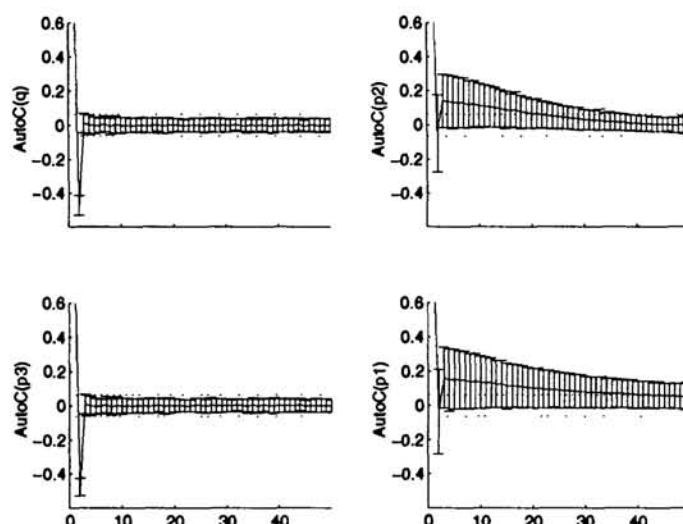

Figure 3: Comparison of autocorrelation functions of the changes in the original observed prices (the upper-left panel), the inventory effect component (the lower-left panel), the regular information component (the upper-right panel) and the surprise information component (the lower-right panel). The results presented are means and standard deviations, and the horizontal dotted lines represent the 95% confidence band. The DEM/USD in September 1995 is divided into 293 sub-sets of 1024 ticks with overlapping of 512 ticks.

(2) The autocorrelation function of original returns reflects only the price changes due to the inventory effect. This further confirms that the existence of short term memory can mislead the analysis of dependence on longer time scales. Subsequently, we can see the usefulness of the multi-effect decomposition. Our empirical results should be viewed as preliminary, since they may depend upon the choice of *True Price* model. Additional studies are ongoing.

## 4 Conclusion and Discussion

We have developed a neural-net-based independent component analysis (ICA) for the multi-effect decomposition of high frequency financial data. Empirical results with foreign exchange rates have demonstrated that the decomposed components are mutually independent. The obtained regular information component has recovered the trending behavior of the intrinsic price movements.

Potential applications for multi-effect decompositions include:
(1) outlier detection and filtering: Filtering techniques for removing various noisy effects and identifying long term trends have been widely studied (see for example Assimakopoulos (1995)). Multi-effect decompositions provide us with an alternative approach. As demonstrated in Section 3, the regular information component can, in most cases, catch relatively stable and longer term trends originally embedded in the price quotes.
(2) devolatilization: Price series are heteroscedastic (Bollerslev, Chou & Kroner 1992). Devolatilization has been widely studied (see, for example, Zhou (1995)). The regular information component obtained from our multi-effect decomposition appears less volatile, and furthermore, its volatility changes more smoothly compared to the original prices.
(3) mixture of local experts modeling: In most cases, one might be interested in only stable, long term trends of price movements. However, the surprise information and inventory effect components are not totally useless. By decomposing the price series into three mutually

independent components, the prices can be modeled by a mixture of local experts (Jacobs, Jordan & Barto 1990), and better modeling performances can be expected.

## Footnotes

[1]This terminology is borrowed from the financial economics literature. For additional properties of high frequency foreign exchange price series, see (Guilaumet, Dacorogna, Dave, Muller, Olsen & Pictet 1994).

## References

Amari, S., Cichocki, A. & Yang, H. (1996), A new learning algorithm for blind signal separation, *in* D. Touretzky, M. Mozer & M. Hasselmo, eds, 'Advances in Neural Information Processing Systems 8', MIT Press: Cambridge, MA.

Assimakopoulos, V. (1995), 'A successive filtering technique for identifying long-term trends', *Journal of Forecasting* **14**, 35–43.

Barlow, H. (1961), Possible principles underlying the transformation of sensory messages, *in* W. Rosenblith, ed., 'Sensory Communication', MIT Press: Cambridge, MA, pp. 217–234.

Bell, A. & Sejnowski, T. (1995), 'An information-maximization approach to blind separation and blind deconvolution', *Neural Computation* **7**(6), 1129–1159.

Bollerslev, T., Chou, R. & Kroner, K. (1992), 'ARCH modelling in finance: A review of the theory and empirical evidence', *Journal of Econometrics* **8**, 5–59.

Cao, X. & Liu, R. (1996), 'General approach to blind source separation', *IEEE Transactions on Signal Processing* **44**(3), 562–569.

Comon, P. (1994), 'Independent component analysis, a new concept?', *Signal Process* **36**, 287–314.

Guilaumet, D., Dacorogna, M., Dave, R., Muller, U., Olsen, R. & Pictet, O. (1994), From the bird's eye to the microscope, a survey of new stylized facts of the intra-daily foreign exchange markets, Technical Report DMG.1994-04-06, Olsen & Associates, Zurich, Switzerland.

Jacobs, R., Jordan, M. & Barto, A. (1990), Task decomposition through competition in a modular connectionist architecture: The what and where vision tasks, Technical Report COINS 90-27, Department of Brain & Cognitive Sciences, Massachusetts Institute of Technology, Cambridge, MA.

Jutten, C. & Herault, J. (1991), 'Blind separation of sources, part I: An adaptive algorithm based on neuromimetic architecture', *Signal Process* **24**(1), 1–10.

Linsker, R. (1989), An application of the principle of maximum information preservation to linear systems, *in* D. Touretzky, ed., 'Advances in Neural Information Processing Systems 1', Morgan Kaufmann Publishers, San Francisco, CA.

Mandelbrot, B. (1963), 'The variation of certain speculative prices', *Journal of Business* **36**, 394–419.

Mandelbrot, B. & Van Ness, J. (1968), 'Fractional Brownian motion, fractional noise, and applications', *SIAM Review* **10**.

Moody, J. & Wu, L. (1995), Statistical analysis and forecasting of high frequency foreign exchange rates, *in* 'The First International Conference on High Frequency Data in Finance', Zurich, Switzerland.

Moody, J. & Wu, L. (1996), What is the 'True Price'? – state space models for high frequency financial data, *in* S. Amari, L. Xu, L. Chan, I. King & K. Leung, eds, 'Progress in Neural Information Processing (Proceedings of ICONIPS*96, Hong Kong)', Springer-Verlag, Singapore, pp. 697–704, Vol.2.

O'Hara, M. (1995), *Market Microstructure Theory*, Blackwell Business.

Pearlmutter, B. A. & Parra, L. (1997), Maximum likelihood blind source separation: a context-sensitive generalization of ICA, *in* M. Mozer, M. Jordan & T. Petsche, eds, 'Advances in Neural Information Processing Systems 9', MIT Press: Cambridge, MA.

Widrow, B., Glover, J., McCool, J., Kaunitz, J., Williams, C., Hearn, R., Zeidler, J., Dong, E. & Goodlin, R. (1975), 'Adaptive noise cancelling: principles and applications', *Proceedings of IEEE* **63**(12), 1692–1716.

Wornell, G. & Oppenheim, A. (1992), 'Estimation of fractal signals from noisy measurements using wavelets', *IEEE Transactions on Signal Processing* **40**(3), 611–623.

Zhou, B. (1995), Forecasting foreign exchange rates series subject to de-volatilization, *in* 'The First International Conference on High Frequency Data in Finance', Zurich, Switzerland.
